# A Low-Power Analog VLSI Visual Collision Detector

**Reid R. Harrison**
Department of Electrical and Computer Engineering
University of Utah
Salt Lake City, UT 84112
*harrison@ece.utah.edu*

## Abstract

We have designed and tested a single-chip analog VLSI sensor that detects imminent collisions by measuring radially expansive optic flow. The design of the chip is based on a model proposed to explain leg-extension behavior in flies during landing approaches. A new elementary motion detector (EMD) circuit was developed to measure optic flow. This EMD circuit models the bandpass nature of large monopolar cells (LMCs) immediately postsynaptic to photoreceptors in the fly visual system. A $16 \times 16$ array of 2-D motion detectors was fabricated on a 2.24 mm $\times$ 2.24 mm die in a standard 0.5-$\mu$m CMOS process. The chip consumes 140 $\mu$W of power from a 5 V supply. With the addition of wide-angle optics, the sensor is able to detect collisions around 500 ms before impact in complex, real-world scenes.

## 1 Introduction

Many animals – from flies to humans – are capable of visually detecting imminent collisions caused either by a rapidly approaching object or self-motion towards an obstacle. Neurons dedicated to this task have been found in the locust [1] and the pigeon [2]. Borst and Bahde have shown that flies use visual information to time the extension of their legs on landing approaches [3].

While several models have been proposed to explain collision detection, the model proposed in [3] is particularly amenable to hardware implementation. The model, shown in Fig. 1, employs a radially-oriented array of motion detectors centered in the direction of flight. As the animal approaches a static object, an expansive optic flow field is produced on the retina. A wide angle field of view is useful since optic flow in the direction of flight will be zero. The response of this radial array of motion detectors is summed and then passed through a leaky integrator (a lowpass filter). If this response exceeds a fixed threshold, an imminent collision is detected and the animal can take evasive action or prepare for a landing. This expansive optic flow model has recently been used to explain landing and collision avoidance responses in the fruit fly [4]. A similar algorithm has been implemented in a traditional CPU for autonomous robot navigation [5]. In this work, we present a single-chip analog VLSI sensor developed to implement this model.

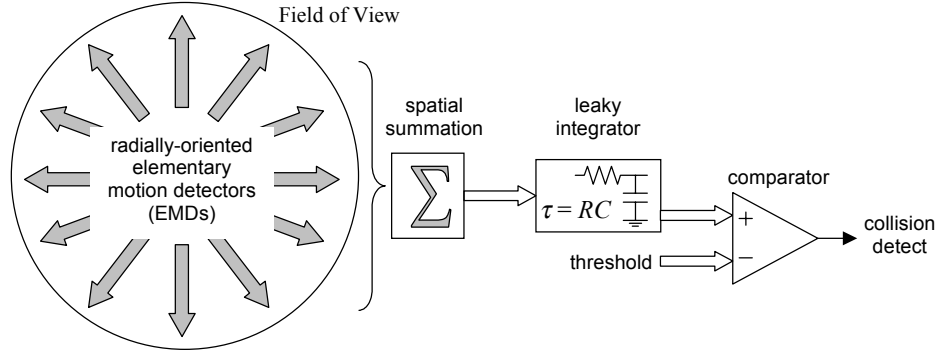

Figure 1: Diagram of collision detection algorithm.

## 2  Elementary Motion Detectors

Our collision detection algorithm uses an array of radially-oriented elementary motion detectors (EMDs) to sense image expansion. Simulations by the author have shown that the structure and properties of the EMDs strongly affect the accuracy of this algorithm [6]. We use an enhanced version of the familiar delay-and-correlate or "Reichardt" EMD first proposed by Hassenstein and Reichardt in the 1950s to explain the optomotor response of beetles [7]. Fig. 2 shows a diagram of the EMD used in our collision sensor.

The first stage of the EMD is photoreception, where light intensity is transduced to a signal $v_{photo}$. Since light intensity is a strictly positive value, the mean intensity of the scene must be subtracted. Since we are interested in motion, it is also advantageous to amplify transient signals.

Suppressing dc illumination and enhancing ac components of photoreceptor signals is a common theme in many biological visual systems. In flies, large monopolar cells (LMCs) directly postsynaptic to photoreceptors exhibit transient biphasic impulse responses approximately 40-200 ms in duration [8], [9]. In the frequency domain, this can be seen as a bandpass filtering operation that attenuates dc signals while amplifying signals in the 2-40 Hz range [9], [10]. In the lateral geniculate nucleus of cats, "lagged" and "non-lagged" cells exhibit transient biphasic impulse responses 200-300 ms in duration and act as bandpass filters amplifying signals in the 1-10 Hz range [11]. This filtering has recently been explained in terms of temporal decorrelation, and can be seen as way of removing redundant information from the photoreceptor signal before further processing [9], [12].

After this "transient enhancement", or temporal decorrelation, the signals are delayed using the phase lag of a lowpass filter. While not a true time delay, the lowpass filter matches data from animal experiments and makes the Reichardt EMD equivalent to the oriented spatiotemporal energy filter proposed by Adelson and Bergen [13]. Before correlating the adjacent delayed and non-delayed signals, we apply a saturating static nonlinearity to each channel. Without such a nonlinearity, the delay-and-correlate EMD exhibits a quadratic dependence on image contrast. In fly tangential neurons, motion responses show a quadratic dependence only at very low contrasts, then quickly become largely independent of image contrast for contrasts above 30%. Egelhaaf and Borst proposed the presence of this nonlinearity in the biological EMD to explain this contrast independence [14]. Functionally, it is necessary to prevent high-contrast edges from dominating the summed output of the EMD array.

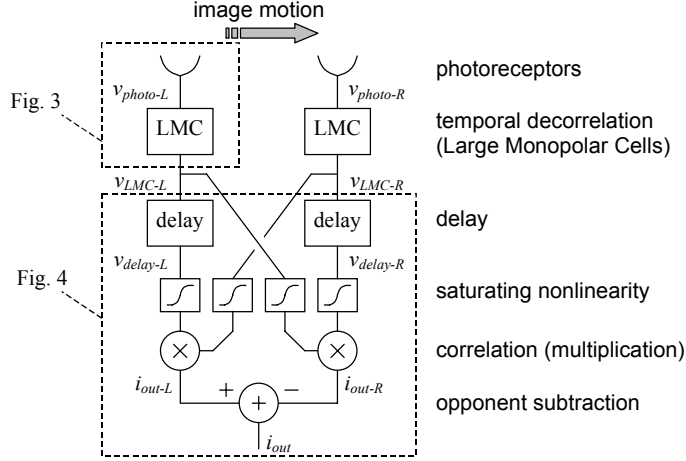

Figure 2: Elaborated delay-and-correlate elementary motion detector (EMD)

After correlation, opponent subtraction produces a strong directionally selective signal that is taken as the output of the EMD. Unlike algorithms that find and track features in an image, the delay-and-correlate EMD does not measure true image velocity independent of the spatial structure of the image. However, recent work has shown that for natural scenes, these Reichardt EMDs give reliable estimates of image velocity [15]. This reliability is improved by the addition of LMC bandpass filters and saturating nonlinearities. Experiments using earlier versions of silicon EMDs have demonstrated the ability of delay-and-correlate motion detectors to work at very low signal-to-noise ratios [16].

## 3   Integrated Circuit Implementation

We adapted the EMD shown in Fig. 2 to a small, low-power CMOS integrated circuit. Fig. 3 shows a schematic of the photoreceptor and LMC bandpass filter. A 35 μm × 35 μm well-substrate photodiode with diode-connected pMOS load converts the diode photocurrent into a voltage $v_{photo}$ that is a logarithmic function of light intensity. A pMOS source follower biased by $I_{SF}$ = 700 pA buffers this signal so that the input capacitance of the LMC circuit does not load the photoreceptor.

The LMC bandpass filter consists of two operational transconductance amplifiers (OTAs) and three capacitors. The OTAs in the circuit are implemented with pMOS differential pairs using diode-connected transistors for source degeneration for extended linear range (see inset, Fig. 3). The transfer function of the LMC circuit is given by

$$\frac{v_{LMC}(s)}{v_{in}(s)} = -A \cdot \frac{N\tau_0 s \cdot (1 - \tau_0 s)}{(\tau_1 s)^2 + \frac{\tau_1 s}{Q} + 1} = -\frac{AN}{\beta} \cdot \frac{\left(1 - \frac{\tau_1}{\beta}s\right)}{\tau_1 s + \frac{1}{Q} + \frac{1}{\tau_1 s}} \tag{1}$$

where

$$\tau_0 = \frac{C}{g_m} \tag{2}$$

$$\beta = \sqrt{N(A+1)(K+1) - N} \approx \sqrt{NAK} \text{ if } A, K \gg 1 \tag{3}$$

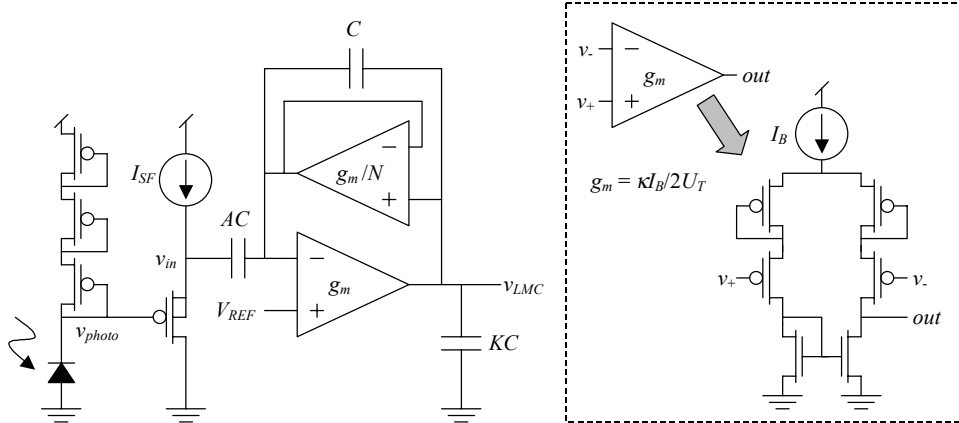

Figure 3: Schematic of photoreceptor/LMC circuit. Detail of operational transconductance amplifier (OTA) shown in inset.

$$\tau_1 = \beta\tau_0 \tag{4}$$

$$Q = \frac{\beta}{(K+N)} \tag{5}$$

The output signal $v_{LMC}$ is centered around $V_{REF}$, a dc voltage which was set to 1.0 V. We sized the capacitors in our circuit to give $A = 20$ and $K = 5$ (with $C = 70$ fF). The transconductance of the lower OTA was set by adjusting its bias current $I_B$:

$$g_m = \frac{\kappa}{(\kappa+1)} \cdot \frac{I_B}{2U_T} \tag{6}$$

where $\kappa$ is the weak inversion slope (typically between 0.6 and 0.9) and $U_T$ is the thermal voltage $kT/q$ (approximately 26 mV at room temperature). We set the bias current in the upper OTA five times smaller to achieve $N = 5$.

As we see from (1), the LMC circuit acts as an ac-coupled bandpass filter centered at $f_1 = 1/2\pi\tau_1$, with a quality factor $Q$ set to 2.5 by capacitor and current ratios. The circuit also has a zero at $\beta f_1$, but since $\beta = 25$ in our circuit, the zero takes effect outside that passband and thus has little practical effect on the filter. We used a bias current of $I_B = 35$ pA in the lower OTA and 7 pA in the upper OTA to center the passband near 20 Hz, which was chosen because it lies in the range of LMC response measured in the fly. This LMC circuit represents a significant improvement over a previous silicon EMD design, which used only a first-order highpass filter to block dc illumination [16]. The LMC circuit presented here allows the designer to adjust the center frequency and $Q$ factor to selectively amplify frequencies present in moving images.

The LMC circuits from each photoreceptor pass their signals to the the delay-and-correlate circuit shown in Fig. 4. The delay is implemented as a first-order lowpass filter. The OTAs in this circuit used two diode-connected transistors in series for extended linear range. The time constant of this filter is given by

$$\tau_{LPF} = \frac{C_{LPF}}{g_{m-LPF}} \tag{7}$$

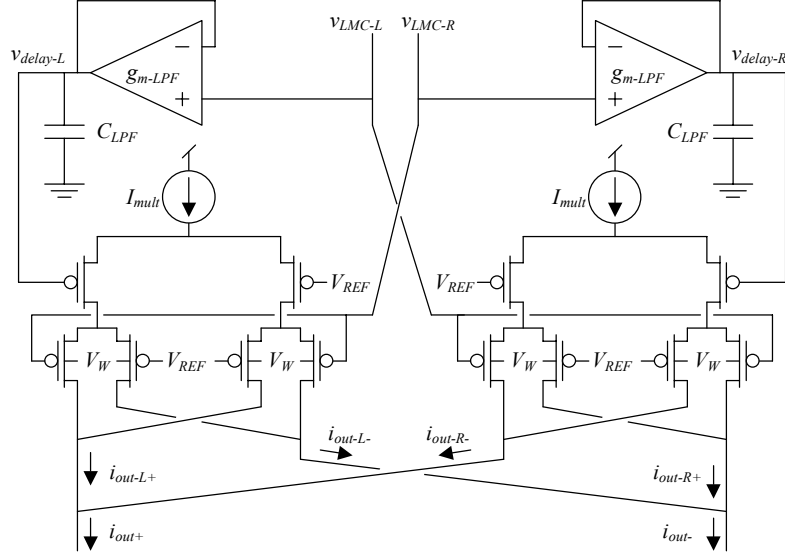

Figure 4: Schematic of delay-and-correlate circuit. OTA-based $g_m$-C filters are used as low-pass filters. Subthreshold CMOS Gilbert multipliers are used for correlation.

We used $C_{LPF}$ = 700 fF and set $\tau_{LPF}$ to around 25 ms, which is in the range of biological motion detectors. This required a bias current of 9 pA for each OTA.

We implemented the correlation function using a CMOS Gilbert multiplier operating in subthreshold [17]. The output currents of the multipliers in Fig. 4 can be expressed as:

$$i_{outL+} - i_{outL-} = I_{mult} \tanh \frac{\kappa(v_{delay-L} - V_{REF})}{2U_T} \tanh \frac{\kappa(v_{LMC-R} - V_{REF})}{2U_T} \tag{8}$$

$$i_{outR+} - i_{outR-} = I_{mult} \tanh \frac{\kappa(v_{delay-R} - V_{REF})}{2U_T} \tanh \frac{\kappa(v_{LMC-L} - V_{REF})}{2U_T} \tag{9}$$

For small differential input voltages, tanh(x) ≈ x and the circuit acts as a linear multiplier. As the input signals grow larger, the tanh nonlinearity dominates and the circuit acts more like a digital exclusive-or gate. We use this inherent circuit nonlinearity as the desired saturating nonlinearity in our EMD model (see Fig. 1). The previous LMC circuit provides sufficient gain to ensure that we are usually operating well *outside* the linear range of the multipliers.

Traditional CMOS Gilbert multipliers require that the dc level of the upper differential input be shifted relative to the dc level of the lower differential input. This is required to keep the transistors in saturation. To avoid the cost in chip area, power consumption, and mismatch associated with level shifters, we introduce a novel circuit modification that allows both the upper and lower differential inputs to operate at the same dc level. We lower the well potential of the lower pMOS transistors from $V_{DD}$ to a dc voltage $V_W$ (see Fig. 4). This lowered well voltage causes the sources of these transistors to operate at a lower potential, which keeps the upper transistors in saturation. We use $V_W$ = 2.5 V in our circuit. (Care must be taken not to make $V_W$ too low, as parasitic source-well-substrate pnp transistors can be activated.)

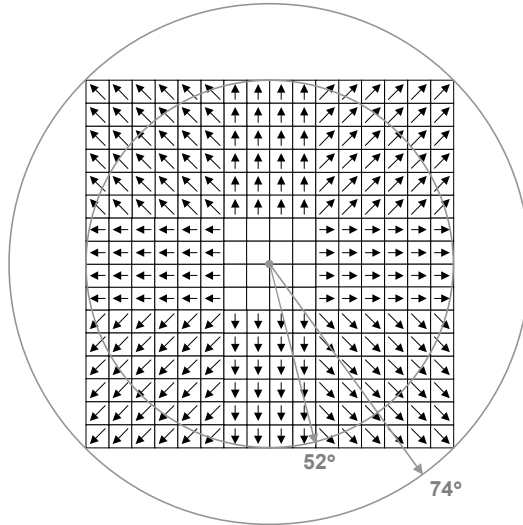

Figure 5: EMD pattern on chip. Ultra-wide-angle optics gave the chip a field of view ranging from ±52° to ±74°.

The output of the Gilbert multiplier is a differential current. The signals from the left and right correlators are easily subtracted by summing their currents appropriately. Similarly, current summation on two global wires is used to sum the motion signals over the entire EMD array.

## 4   Experimental Results

We fabricated a 16 × 16 EMD array in a 0.5-µm 2-poly, 3-metal standard CMOS process. The 2.24 mm × 2.24 mm die contained a 17 × 17 array of "pixels," each measuring 100 µm × 100 µm. Each pixel contained a photoreceptor, LMC circuit, lowpass "delay" filter, and four correlators. These correlators were used to implement two independent EMDs: a vertical motion detector connected to the pixel below and a horizontal motion detector connected to the pixel to the right. The output signals from a subset of the EMDs representing radial outward motion were connected to two global wires, giving a differential current signal that was taken off chip on two pins.

Fig. 5 shows the EMDs that were summed to produce the global radial motion signal. Diagonally-oriented EMDs were derived from the sum of a horizontal and a vertical EMD. The center 4 × 4 pixels were ignored, as motion near the center of the field of view is typically very small in collision situations. We used custom-built ultra-wide-angle optics to give the chip a field of view ranging from ±52° at the sides to ±74° at the corners. Simulations revealed that a field of view of around ±60° was necessary for reasonable performance using this algorithm [6].

Before testing the array, we characterized an individual LMC circuit configured to have a voltage input $v_{photo}$ provided from off chip using a function generator. We provided a 1.4 Hz, 100 mVpp square wave and observed the LMC circuit output. As shown in Fig. 6a, the LMC circuit exhibits a transient oscillatory step response similar to its biological counterpart. Using a spectrum analyzer, we measured the transfer function of the circuit (see Fig. 6b). The LMC circuit acts as a bandpass filter centered at 19 Hz, with a measured $Q$ of 2.3.

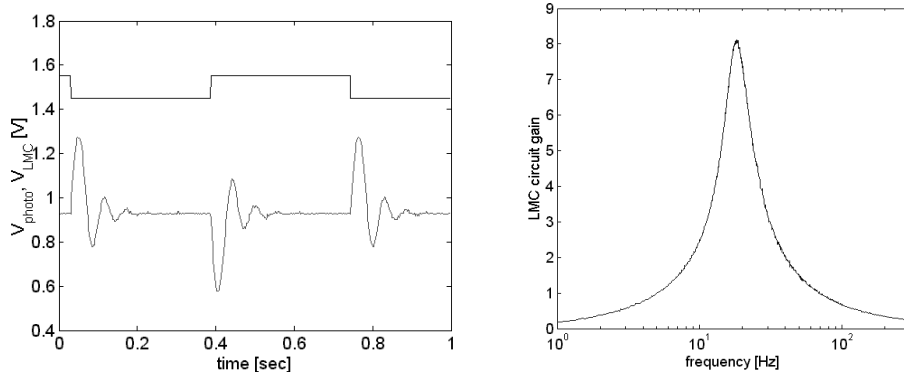

Figure 6: Measurement of LMC circuit performance. (a) Step response of LMC circuit. (b) Frequency tuning of LMC circuit.

The entire chip consumed 140 μW of power. Most of this was consumed by peripheral biasing circuits; the 17 × 17 pixel array used only 5.2 μW (18 nW per pixel). To test the complete collision detection chip, we implemented the leaky integrator ($\tau_{leak}$ = 50 ms) and comparator from Fig. 1 using off-chip components. In future implementations, these circuits could be built on chip using little power.

We tested the chip by mounting it on a small motorized vehicle facing forward with the lens centered 11 cm above the floor. The vehicle traveled in a straight path at 28 cm/s. Fig. 7 shows the output from the leaky integrator as the chip moves across the floor and collides with the center of a 38 cm × 38 cm trash can in our lab. The peak response of the chip occurs approximately 500 ms before contact, which corresponds to a distance of 14 cm. At this point, the edges of the trash can subtend an angle of 54°. After this point, the edges of the can move beyond the chip's field of view, and the response decays rapidly. The rebound in response observed in the last 100 ms may be due to the chip seeing the expanding shadow cast by its own lens on the side of the can just before contact.

## 5  Conclusions

The response of our chip, which peaks and then collapses before impact, is similar to activity patterns observed in the LGMD neuron in locusts [1] and η neurons in pigeons [2] during simulated collisions. While more complex models positing the measurement of true image velocity and object size have been used to explain this peculiar time course [1], we observe that a simple model integrating the output of a radial EMD array gives qualitatively similar responses.

We have demonstrated that this model of collision detection can be implemented in a small, low-power, single-chip sensor. Further testing of the chip on mobile platforms should better characterize its performance.

**Acknowledgments**

This work was partially supported by a contract from the Naval Air Warfare Center, China Lake, CA.

**References**

[1] F. Gabbiani, H.G. Krapp, and G. Laurent, "Computation of object approach by a wide-field, motion-sensitive neuron," *J. Neurosci.* **19:**1122-1141, 1999.

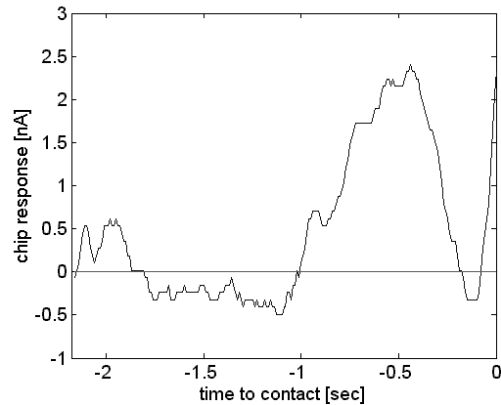

Figure 7: Measured output of collision detection chip.

[2] H. Sun and B.J. Frost, "Computation of different optical variables of looming objects in pigeon nucleus rotundus neurons," *Nature Neurosci.* **1:**296-303, 1998.

[3] A. Borst and S. Bahde, "Visual information processing in the fly's landing system," *J. Comp. Physiol. A* **163:**167-173, 1988.

[4] L.T. Tammero and M.H. Dickinson, "Collision-avoidance and landing responses are mediated by separate pathways in the fruit fly, *Drosophila melanogaster*," *J. Exp. Biol.***205:** 2785-2798, 2002.

[5] A.P. Duchon, W.H. Warren, and L.P. Kaelbling, "Ecological robotics," *Adaptive Behavior* **6:**473-507, 1998.

[6] R.R. Harrison, "An algorithm for visual collision detection in real-world scenes," submitted to *NIPS 2003.*

[7] B. Hassenstein and W. Reichardt, "Systemtheoretische Analyse der Zeit-, Reihenfolgen-, und Vorzeichenauswertung bei der Bewegungsperzeption des Rüsselkäfers *Chlorophanus,*" *Z. Naturforch.* **11b:**513-524, 1956.

[8] S.B. Laughlin, "Matching coding, circuits, cells, and molecules to signals – general principles of retinal design in the fly's eye," *Progress in Ret. Eye Research* **13:**165-196, 1994.

[9] J.H. van Hateren, "Theoretical predictions of spatiotemporal receptive fields of fly LMCs, and experimental validation," *J. Comp. Physiol. A* **171:**157-170, 1992.

[10] J.H. van Hateren, "Processing of natural time series of intensities by the visual system of the blowfly," *Vision Res.* **37:**3407-3416, 1997.

[11] A.B. Saul and A.L. Humphrey, "Spatial and temporal response properties of lagged and nonlagged cells in cat lateral geniculate nucleus," *J. Neurophysiol.* **64:**206-224, 1990.

[12] D.W. Dong and J.J. Atick, "Temporal decorrelation: a theory of lagged and nonlagged responses in the lateral geniculate nucleus," *Network* **6:**159-178, 1995.

[13] E.H. Adelson and J.R. Bergen, "Spatiotemporal energy models for the perception of motion," *J. Opt. Soc. Am. A* **2:**284-299, 1985.

[14] M. Egelhaaf and A. Borst, "Transient and steady-state response properties of movement detectors," *J. Opt. Soc. Am. A* **6:**116-127, 1989.

[15] R.O. Dror, D.C. O'Carroll, and S.B. Laughlin, "Accuracy of velocity estimation by Reichardt correlators," *J. Opt. Soc. Am. A* **18:**241-252, 2001.

[16] R.R. Harrison and C. Koch, "A robust analog VLSI Reichardt motion sensor," *Analog Integrated Circuits and Signal Processing* **24:**213-229, 2000.

[17] C. Mead, *Analog VLSI and Neural Systems,* Reading, MA: Addison-Wesley, 1989.
